# LSTM CAN SOLVE HARD LONG TIME LAG PROBLEMS

**Sepp Hochreiter**
Fakultät für Informatik
Technische Universität München
80290 München, Germany

**Jürgen Schmidhuber**
IDSIA
Corso Elvezia 36
6900 Lugano, Switzerland

## Abstract

Standard recurrent nets cannot deal with long minimal time lags between relevant signals. Several recent NIPS papers propose alternative methods. We first show: problems used to promote various previous algorithms can be solved more quickly by random weight guessing than by the proposed algorithms. We then use LSTM, our own recent algorithm, to solve a hard problem that can neither be quickly solved by random search nor by any other recurrent net algorithm we are aware of.

## 1 TRIVIAL PREVIOUS LONG TIME LAG PROBLEMS

Traditional recurrent nets fail in case of long minimal time lags between input signals and corresponding error signals [7, 3]. Many recent papers propose alternative methods, e.g., [16, 12, 1, 5, 9]. For instance, Bengio et al. investigate methods such as simulated annealing, multi-grid random search, time-weighted pseudo-Newton optimization, and discrete error propagation [3]. They also propose an EM approach [1]. Quite a few papers use variants of the "2-sequence problem" (and "latch problem") to show the proposed algorithm's superiority, e.g. [3, 1, 5, 9]. Some papers also use the "parity problem", e.g., [3, 1]. Some of Tomita's [18] grammars are also often used as benchmark problems for recurrent nets [2, 19, 14, 11].

**Trivial versus non-trivial tasks.** By our definition, a "trivial" task is one that can be solved quickly by random search (RS) in weight space. RS works as follows: *REPEAT randomly initialize the weights and test the resulting net on a training set UNTIL solution found.*

**Random search (RS) details.** In all our RS experiments, we randomly initialize weights in [-100.0,100.0]. Binary inputs are -1.0 (for 0) and 1.0 (for 1). Targets are either 1.0 or 0.0. All activation functions are logistic sigmoid in [0.0,1.0]. We use two architectures (A1, A2) suitable for many widely used "benchmark" problems: A1 is a fully connected net with 1 input, 1 output, and $n$ biased hidden units. A2 is like A1 with $n = 10$, but less densely connected: each hidden unit sees the input unit, the output unit, and itself; the output unit sees all other units; all units are biased. All activations are set to 0 at each sequence begin. We will indicate where we also use different architectures of other authors. All sequence lengths are randomly chosen between 500 and 600 (most other authors facilitate their problems by using much shorter training/test sequences). The "benchmark" problems always require to classify two types of sequences. Our training set consists of 100 sequences, 50 from class 1 (target 0) and 50 from class 2 (target 1). Correct sequence classification is defined as "absolute error at sequence end below 0.1". We stop the search once a random weight matrix correctly classifies all training sequences. Then we test on the test set (100 sequences). All results below are averages of 10 trials. **In all our simulations below, RS finally classified all test set sequences correctly; average final absolute test set errors were always below 0.001 — in most cases below 0.0001.**

**"2-sequence problem"** (and "latch problem") [3, 1, 9]. The task is to observe and classify input sequences. There are two classes. There is only one input unit or input line. Only the first N real-valued sequence elements convey relevant information about the class. Sequence elements at positions $t > N$ (we use $N = 1$) are generated by a Gaussian with mean zero and variance 0.2. The first sequence element is 1.0 (-1.0) for class 1 (2). Target at sequence end is 1.0 (0.0) for class 1 (2) (the latch problem is a simple version of the 2-sequence problem that allows for input tuning instead of weight tuning).

**Bengio et al.'s results.** For the 2-sequence problem, the best method among the six tested by Bengio et al. [3] was multigrid random search (sequence lengths 50 — 100; N and stopping criterion undefined), which solved the problem after 6,400 sequence presentations, with final classification error 0.06. In more recent work, Bengio and Frasconi reported that an EM-approach [1] solves the problem within 2,900 trials.

**RS results.** RS with architecture A2 (A1, $n = 1$) solves the problem within only 718 (1247) trials on average. Using an architecture with only 3 parameters (as in Bengio et al.'s architecture for the latch problem [3]), the problem was solved within only 22 trials on average, due to tiny parameter space. According to our definition above, the problem is trivial. RS outperforms Bengio et al.'s methods in every respect: (1) many fewer trials required, (2) much less computation time per trial. Also, in most cases (3) the solution quality is better (less error).

It should be mentioned, however, that different input representations and different types of noise may lead to worse RS performance (Yoshua Bengio, personal communication, 1996).

**"Parity problem".** The parity task [3, 1] requires to classify sequences with several 100 elements (only 1's or -1's) according to whether the number of 1's is even or odd. The target at sequence end is 1.0 for odd and 0.0 for even.

**Bengio et al.'s results.** For sequences with only 25-50 steps, among the six methods tested in [3] only simulated annealing was reported to achieve final classification error of 0.000 (within about 810,000 trials — the authors did not mention the precise stopping criterion). A method called "discrete error BP" took about 54,000 trials to achieve final classification error 0.05. In more recent work [1], for sequences with 250-500 steps, their EM-approach took about 3,400 trials to achieve final classification error 0.12.

**RS results.** RS with A1 ($n = 1$) solves the problem within only 2906 trials on average. RS with A2 solves it within 2797 trials. We also ran another experiment with architecture A2, but without self-connections for hidden units. RS solved the problem within 250 trials on average.

Again it should be mentioned that different input representations and noise types may lead to worse RS performance (Yoshua Bengio, personal communication, 1996).

**Tomita grammars.** Many authors also use Tomita's grammars [18] to test their algorithms. See, e.g., [2, 19, 14, 11, 10]. Since we already tested parity problems above, we now focus on a few "parity-free" Tomita grammars (nr.s #1, #2, #4). Previous work facilitated the problems by restricting sequence length. E.g., in [11], maximal test (training) sequence length is 15 (10). Reference [11] reports the number of sequences required for convergence (for various first and second order nets with 3 to 9 units): Tomita #1: 23,000 – 46,000; Tomita #2: 77,000 – 200,000; Tomita #4: 46,000 – 210,000. RS, however, clearly outperforms the methods in [11]. The average results are: Tomita #1: 182 (A1, $n = 1$) and 288 (A2), Tomita #2: 1,511 (A1, $n = 3$) and 17,953 (A2), Tomita #4: 13,833 (A1, $n = 2$) and 35,610 (A2).

**Non-trivial tasks / Outline of remainder.** Solutions of non-trivial tasks are sparse in weight space. They require either many free parameters (e.g., input weights) or high weight precision, such that RS becomes infeasible. To solve such tasks we need a novel method called "Long Short-Term Memory", or LSTM for short [8]. Section 2 will briefly review LSTM. Section 3 will show results on a task that cannot be solved at all by any other recurrent net learning algorithm we are aware of. The task involves distributed, high-precision, continuous-valued representations and long minimal time lags — there are no short time lag training exemplars facilitating learning.

## 2 LONG SHORT-TERM MEMORY

**Memory cells and gate units: basic ideas.** LSTM's basic unit is called a memory cell. Within each memory cell, there is a linear unit with a fixed-weight self-connection (compare Mozer's time constants [12]). This enforces constant, non-exploding, non-vanishing error flow within the memory cell. A multiplicative *input gate unit* learns to protect the constant error flow within the memory cell from perturbation by irrelevant inputs. Likewise, a multiplicative *output gate unit* learns to protect other units from perturbation by currently irrelevant memory contents stored in the memory cell. The gates learn to open and close access to constant error flow. *Why is constant error flow important?* For instance, with conventional "back-prop through time" (BPTT, e.g., [20]) or RTRL (e.g., [15]), error signals "flowing backwards in time" tend to vanish: the temporal evolution of the backpropagated

error exponentially depends on the size of the weights. For the first theoretical error flow analysis see [7]. See [3] for a more recent, independent, essentially identical analysis.

**LSTM details.** In what follows, $w_{uv}$ denotes the weight on the connection from unit $v$ to unit $u$. $net_u(t), y^u(t)$ are net input and activation of unit $u$ (with activation function $f_u$) at time $t$. For all non-input units that aren't memory cells (e.g. output units), we have $y^u(t) = f_u(net_u(t))$, where $net_u(t) = \sum_v w_{uv}y^v(t-1)$. The $j$-th memory cell is denoted $c_j$. Each memory cell is built around a central linear unit with a fixed self-connection (weight 1.0) and identity function as activation function (see definition of $s_{c_j}$ below). In addition to $net_{c_j}(t) = \sum_u w_{c_j u}y^u(t-1)$, $c_j$ also gets input from a special unit $out_j$ (the "output gate"), and from another special unit $in_j$ (the "input gate"). $in_j$'s activation at time $t$ is denoted by $y^{in_j}(t)$. $out_j$'s activation at time $t$ is denoted by $y^{out_j}(t)$. $in_j$, $out_j$ are viewed as ordinary hidden units. We have $y^{out_j}(t) = f_{out_j}(net_{out_j}(t)), y^{in_j}(t) = f_{in_j}(net_{in_j}(t))$, where $net_{out_j}(t) = \sum_u w_{out_j u}y^u(t-1)$, $net_{in_j}(t) = \sum_u w_{in_j u}y^u(t-1)$. The summation indices $u$ may stand for input units, gate units, memory cells, or even conventional hidden units if there are any (see also paragraph on "network topology" below). All these different types of units may convey useful information about the current state of the net. For instance, an input gate (output gate) may use inputs from other memory cells to decide whether to store (access) certain information in its memory cell. There even may be recurrent self-connections like $w_{c_j c_j}$. It is up to the user to define the network topology. At time $t$, $c_j$'s output $y^{c_j}(t)$ is computed in a sigma-pi-like fashion: $y^{c_j}(t) = y^{out_j}(t)h(s_{c_j}(t))$, where

$$s_{c_j}(0) = 0, s_{c_j}(t) = s_{c_j}(t-1) + y^{in_j}(t)g\left(net_{c_j}(t)\right) \text{ for } t > 0.$$

The differentiable function $g$ scales $net_{c_j}$. The differentiable function $h$ scales memory cell outputs computed from the internal state $s_{c_j}$.

**Why gate units?** $in_j$ controls the error flow to memory cell $c_j$'s input connections $w_{c_j u}$. $out_j$ controls the error flow from unit $j$'s output connections. Error signals trapped within a memory cell *cannot* change – but different error signals flowing into the cell (at different times) via its output gate may get superimposed. The output gate will have to learn *which* errors to trap in its memory cell, by appropriately scaling them. Likewise, the input gate will have to learn when to release errors. Gates open and close access to constant error flow.

**Network topology.** There is one input, one hidden, and one output layer. The fully self-connected hidden layer contains memory cells and corresponding gate units (for convenience, we refer to both memory cells and gate units as hidden units located in the hidden layer). The hidden layer may also contain "conventional" hidden units providing inputs to gate units and memory cells. All units (except for gate units) in all layers have directed connections (serve as inputs) to all units in higher layers.

**Memory cell blocks.** $S$ memory cells sharing one input gate and one output gate form a "memory cell block of size $S$". They can facilitate information storage.

**Learning with excellent computational complexity** — see details in appendix of [8]. We use a variant of RTRL which properly takes into account the altered (sigma-pi-like) dynamics caused by input and output gates. However, to ensure constant error backprop, like with truncated BPTT [20], errors arriving at "memory

cell net inputs" (for cell $c_j$, this includes $net_{c_j}$, $net_{in_j}$, $net_{out_j}$) do not get propagated back further in time (although they *do* serve to change the incoming weights). Only within memory cells, errors are propagated back through previous internal states $s_{c_j}$. This enforces constant error flow within memory cells. Thus only the derivatives $\frac{\partial s_{c_j}}{\partial w_{il}}$ need to be stored and updated. Hence, **the algorithm is very efficient,** and LSTM's update complexity per time step is excellent in comparison to other approaches such as RTRL: given $n$ units and a fixed number of output units, LSTM's update complexity per time step is at most $O(n^2)$, just like BPTT's.

## 3   EXPERIMENT: ADDING PROBLEM

Our previous experimental comparisons (on widely used benchmark problems) with RTRL (e.g., [15]; results compared to the ones in [17]), Recurrent Cascade-Correlation [6], Elman nets (results compared to the ones in [4]), and Neural Sequence Chunking [16], demonstrated that LSTM leads to many more successful runs than its competitors, and learns much faster [8]. The following task, though, is more difficult than the above benchmark problems: it cannot be solved at all in reasonable time by RS (we tried various architectures) nor any other recurrent net learning algorithm we are aware of (see [13] for an overview). The experiment will show that LSTM can solve non-trivial, complex long time lag problems involving distributed, high-precision, continuous-valued representations.

**Task.** Each element of each input sequence is a pair consisting of two components. The first component is a real value randomly chosen from the interval $[-1, 1]$. The second component is either 1.0, 0.0, or -1.0, and is used as a marker: at the end of each sequence, the task is to output the sum of the first components of those pairs that are *marked* by second components equal to 1.0. The value $T$ is used to determine average sequence length, which is a randomly chosen integer between $T$ and $T + \frac{T}{10}$. With a given sequence, exactly two pairs are marked as follows: we first randomly select and mark one of the first ten pairs (whose first component is called $X_1$). Then we randomly select and mark one of the first $\frac{T}{2} - 1$ still unmarked pairs (whose first component is called $X_2$). The second components of the remaining pairs are zero except for the first and final pair, whose second components are -1 ($X_1$ is set to zero in the rare case where the *first* pair of the sequence got marked). An error signal is generated only at the sequence end: the target is $0.5 + \frac{X_1 + X_2}{4.0}$ (the sum $X_1 + X_2$ scaled to the interval $[0, 1]$). A sequence was processed correctly if the absolute error at the sequence end is below 0.04.

**Architecture.** We use a 3-layer net with 2 input units, 1 output unit, and 2 memory cell blocks of size 2 (a cell block size of 1 works well, too). The output layer receives connections only from memory cells. Memory cells/ gate units receive inputs from memory cells/gate units (fully connected hidden layer). Gate units ($f_{in_j}$, $f_{out_j}$) and output units are sigmoid in $[0, 1]$. $h$ is sigmoid in $[-1, 1]$, and $g$ is sigmoid in $[-2, 2]$.

**State drift versus initial bias.** Note that the task requires to store the precise values of real numbers for long durations — the system must learn to protect memory cell contents against even minor "internal state drifts". Our simple but highly effective way of solving drift problems at the beginning of learning is to initially bias the input gate $in_j$ towards zero. *There is no need for fine tuning initial bias:* with

sigmoid logistic activation functions, the precise initial bias hardly matters because vastly different initial bias values produce almost the same near-zero activations. In fact, the system itself learns to generate the most appropriate input gate bias. To study the significance of the drift problem, we bias all non-input units, thus artificially inducing internal state drifts. Weights (including bias weights) are randomly initialized in the range $[-0.1, 0.1]$. The first (second) input gate bias is initialized with $-3.0$ ($-6.0$) (recall that the precise initialization values hardly matters, as confirmed by additional experiments).

**Training / Testing.** The learning rate is 0.5. Training examples are generated on-line. Training is stopped if the average training error is below 0.01, and the 2000 most recent sequences were processed correctly (see definition above).

**Results.** With a test set consisting of 2560 randomly chosen sequences, the average test set error was always below 0.01, and there were never more than 3 incorrectly processed sequences. The following results are means of 10 trials: For $T = 100$ ($T = 500$, $T = 1000$), training was stopped after 74,000 (209,000; 853,000) training sequences, and then only 1 (0, 1) of the test sequences was not processed correctly. For $T = 1000$, the number of required training examples varied between 370,000 and 2,020,000, exceeding 700,000 in only 3 cases.

The experiment demonstrates even for very long minimal time lags: (1) LSTM is able to work well with distributed representations. (2) LSTM is able to perform calculations involving *high-precision, continuous* values. Such tasks are impossible to solve within reasonable time by other algorithms: the main problem of gradient-based approaches (including TDNN, pseudo Newton) is their inability to deal with very long minimal time lags (vanishing gradient). A main problem of "global" and "discrete" approaches (RS, Bengio's and Frasconi's EM-approach, discrete error propagation) is their inability to deal with high-precision, continuous values.

**Other experiments.** In [8] LSTM is used to solve numerous additional tasks that cannot be solved by other recurrent net learning algorithm we are aware of. For instance, LSTM can extract information conveyed by the temporal order of widely separated inputs. LSTM also can learn real-valued, conditional expectations of strongly delayed, noisy targets, given the inputs.

**Conclusion.** For non-trivial tasks (where RS is infeasible), we recommend LSTM.

# 4   ACKNOWLEDGMENTS

This work was supported by *DFG grant SCHM 942/3-1* from "Deutsche Forschungsgemeinschaft".

# References

[1] Y. Bengio and P. Frasconi. Credit assignment through time: Alternatives to backpropagation. In J. D. Cowan, G. Tesauro, and J. Alspector, editors, *Advances in Neural Information Processing Systems 6*, pages 75–82. San Mateo, CA: Morgan Kaufmann, 1994.

[2] Y. Bengio and P. Frasconi. An input output HMM architecture. In G. Tesauro, D. S. Touretzky, and T. K. Leen, editors, *Advances in Neural Information Processing Systems 7*, pages 427–434. MIT Press, Cambridge MA, 1995.

[3] Y. Bengio, P. Simard, and P. Frasconi. Learning long-term dependencies with gradient descent is difficult. *IEEE Transactions on Neural Networks*, 5(2):157–166, 1994.

[4] A. Cleeremans, D. Servan-Schreiber, and J. L. McClelland. Finite-state automata and simple recurrent networks. *Neural Computation*, 1:372–381, 1989.

[5] S. El Hihi and Y. Bengio. Hierarchical recurrent neural networks for long-term dependencies. In *Advances in Neural Information Processing Systems 8*, 1995. to appear.

[6] S. E. Fahlman. The recurrent cascade-correlation learning algorithm. In R. P. Lippmann, J. E. Moody, and D. S. Touretzky, editors, *Advances in Neural Information Processing Systems 3*, pages 190–196. San Mateo, CA: Morgan Kaufmann, 1991.

[7] J. Hochreiter. Untersuchungen zu dynamischen neuronalen Netzen. Diploma thesis, Institut für Informatik, Lehrstuhl Prof. Brauer, Technische Universität München, 1991. See www7.informatik.tu-muenchen.de/~hochreit.

[8] S. Hochreiter and J. Schmidhuber. Long short-term memory. Technical Report FKI-207-95, Fakultät für Informatik, Technische Universität München, 1995. Revised 1996 (see www.idsia.ch/~juergen, www7.informatik.tu-muenchen.de/~hochreit).

[9] T. Lin, B. G. Horne, P. Tino, and C. L. Giles. Learning long-term dependencies is not as difficult with NARX recurrent neural networks. Technical Report UMIACS-TR-95-78 and CS-TR-3500, Institute for Advanced Computer Studies, University of Maryland, College Park, MD 20742, 1995.

[10] P. Manolios and R. Fanelli. First-order recurrent neural networks and deterministic finite state automata. *Neural Computation*, 6:1155–1173, 1994.

[11] C. B. Miller and C. L. Giles. Experimental comparison of the effect of order in recurrent neural networks. *International Journal of Pattern Recognition and Artificial Intelligence*, 7(4):849–872, 1993.

[12] M. C. Mozer. Induction of multiscale temporal structure. In J. E. Moody, S. J. Hanson, and R. P. Lippman, editors, *Advances in Neural Information Processing Systems 4*, pages 275–282. San Mateo, CA: Morgan Kaufmann, 1992.

[13] B. A. Pearlmutter. Gradient calculations for dynamic recurrent neural networks: A survey. *IEEE Transactions on Neural Networks*, 6(5):1212–1228, 1995.

[14] J. B. Pollack. The induction of dynamical recognizers. *Machine Learning*, 7:227–252, 1991.

[15] A. J. Robinson and F. Fallside. The utility driven dynamic error propagation network. Technical Report CUED/F-INFENG/TR.1, Cambridge University Engineering Department, 1987.

[16] J. H. Schmidhuber. Learning complex, extended sequences using the principle of history compression. *Neural Computation*, 4(2):234–242, 1992.

[17] A. W. Smith and D. Zipser. Learning sequential structures with the real-time recurrent learning algorithm. *International Journal of Neural Systems*, 1(2):125–131, 1989.

[18] M. Tomita. Dynamic construction of finite automata from examples using hill-climbing. In *Proceedings of the Fourth Annual Cognitive Science Conference*, pages 105–108. Ann Arbor, MI, 1982.

[19] R. L. Watrous and G. M. Kuhn. Induction of finite-state automata using second-order recurrent networks. In J. E. Moody, S. J. Hanson, and R. P. Lippman, editors, *Advances in Neural Information Processing Systems 4*, pages 309–316. San Mateo, CA: Morgan Kaufmann, 1992.

[20] R. J. Williams and J. Peng. An efficient gradient-based algorithm for on-line training of recurrent network trajectories. *Neural Computation*, 4:491–501, 1990.

